# Structured Learning with Approximate Inference

**Alex Kulesza and Fernando Pereira**[*]
Department of Computer and Information Science
University of Pennsylvania
{kulesza, pereira}@cis.upenn.edu

## Abstract

In many structured prediction problems, the highest-scoring labeling is hard to compute exactly, leading to the use of approximate inference methods. However, when inference is used in a learning algorithm, a good approximation of the score may not be sufficient. We show in particular that learning can fail even with an approximate inference method with rigorous approximation guarantees. There are two reasons for this. First, approximate methods can effectively reduce the expressivity of an underlying model by making it impossible to choose parameters that reliably give good predictions. Second, approximations can respond to parameter changes in such a way that standard learning algorithms are misled. In contrast, we give two positive results in the form of learning bounds for the use of LP-relaxed inference in structured perceptron and empirical risk minimization settings. We argue that without understanding combinations of inference and learning, such as these, that are appropriately compatible, learning performance under approximate inference cannot be guaranteed.

## 1  Introduction

Structured prediction models commonly involve complex inference problems for which finding exact solutions is intractable [1]. There are two ways to address this difficulty. Directly, models used in practice can be restricted to those for which inference is feasible, such as conditional random fields on trees [2] or associative Markov networks with binary labels [3]. More generally, however, efficient but approximate inference procedures have been devised that apply to a wide range of models, including loopy belief propagation [4, 5], tree-reweighted message passing [6], and linear programming relaxations [7, 3], all of which give efficient approximate predictions for graphical models of arbitrary structure.

Since some form of inference is the dominant subroutine for all structured learning algorithms, it is natural to see good approximate inference techniques as solutions to the problem of tractable *learning* as well. A number of authors have taken this approach, using inference approximations as drop-in replacements during training, often with empirical success [3, 8]. And yet there has been little theoretical analysis of the relationship between approximate inference and reliable learning.

We demonstrate with two counterexamples that the characteristics of approximate inference algorithms relevant for learning can be distinct from those, such as approximation guarantees, that make them appropriate for prediction. First, we show that approximations can reduce the expressivity of a model, making previously simple concepts impossible to implement and hence to learn, even though inference meets an approximation guarantee. Second, we show that standard learning algorithms can be led astray by inexact inference, failing to find valid model parameters. It is therefore crucial to choose compatible inference and learning procedures.

---

[*]This work is based on research supported by NSF ITR IIS 0428193.

With these considerations in mind, we prove that LP-relaxation-based approximate inference procedures are compatible with the structured perceptron [9] as well as empirical risk minimization with a margin criterion using the PAC-Bayes framework [10, 11].

## 2 Setting

Given a scoring model $S(\mathbf{y}|\mathbf{x})$ over candidate labelings $\mathbf{y}$ for input $\mathbf{x}$, exact Viterbi inference is the computation of the optimal labeling

$$h(\mathbf{x}) = \arg\max_{\mathbf{y}} \; S(\mathbf{y}|\mathbf{x}) \;. \tag{1}$$

In a prediction setting, the goal of approximate inference is to compute efficiently a prediction with the highest possible score. However, in learning a tight relationship between the scoring model and true utility cannot be assumed; after all, learning seeks to find such a relationship. Instead, we assume a fixed loss function $\mathcal{L}(\mathbf{y}|\mathbf{x})$ that measures the true cost of predicting $\mathbf{y}$ given $\mathbf{x}$, a distribution $D$ over inputs $\mathbf{x}$, and a parameterized scoring model $S_\theta(\mathbf{y}|\mathbf{x})$ with associated optimal labeling function $h_\theta$ and inference algorithm $\mathcal{A}_\theta$. Exact inference implies $\mathcal{A}_\theta = h_\theta$. Learning seeks the risk minimizer:

$$\theta^* = \arg\min_{\theta} \; E_{\mathbf{x}\sim D}\left[\mathcal{L}(\mathcal{A}_\theta(\mathbf{x})|\mathbf{x})\right] \;. \tag{2}$$

Successful learning, then, requires two things: the existence of $\theta$ for which risk is suitably low, and the ability to find such $\theta$ efficiently. In this work we consider the impact of approximate inference on both criteria. We model our examples as pairwise Markov random fields (MRFs) defined over a graph $G = (V, E)$ with probabilistic scoring model

$$P(\mathbf{y}|\mathbf{x}) \propto \prod_{i\in V} \psi_i(y_i|\mathbf{x}) \prod_{ij\in E} \psi_{ij}(y_i, y_j|\mathbf{x}) \;, \tag{3}$$

where $\psi_i(y_i|\mathbf{x})$ and $\psi_{ij}(y_i, y_j|\mathbf{x})$ are positive potentials. For learning, we use log-linear potentials $\psi_i(y_i|\mathbf{x}) = \exp(\mathbf{w}\cdot\mathbf{f}(\mathbf{x}, y_i))$ assuming a feature function $\mathbf{f}(\cdot)$ and parameter vector $\mathbf{w}$. Since MRFs are probabilistic, we also refer to Viterbi inference as maximum *a posteriori* (MAP) inference.

## 3 Algorithmic separability

The existence of suitable model parameters $\theta$ is captured by the standard notion of separability.

**Definition 1.** *A distribution $D$ (which can be empirical) is* **separable** *with respect to a model $S_\theta(\mathbf{y}|\mathbf{x})$ and loss $\mathcal{L}(\mathbf{y}|\mathbf{x})$ if there exists $\theta$ such that $E_{\mathbf{x}\sim D}\left[\mathcal{L}(h_\theta(\mathbf{x}), \mathbf{x})\right] = 0$*[1].

However, approximate inference may not be able to match exactly the separating hypothesis $h_\theta$. We need a notion of separability that takes into account the (approximate) inference algorithm.

**Definition 2.** *A distribution $D$ is* **algorithmically separable** *with respect to parameterized inference algorithm $\mathcal{A}_\theta$ and loss $\mathcal{L}(\mathbf{y}|\mathbf{x})$ if there exists $\theta$ such that $E_{\mathbf{x}\sim D}\left[\mathcal{L}(\mathcal{A}_\theta(\mathbf{x}), \mathbf{x})\right] = 0$.*

While separability characterizes data distributions with respect to models, algorithmic separability characterizes data distributions with respect to inference algorithms. Note that algorithmic separability is more general than standard separability for any decidable model, since we can design an (inefficient) algorithm $\mathcal{A}_\theta(\mathbf{x}) = h_\theta(\mathbf{x})$[2]. However, we show by counterexample that even algorithms with provable approximation guarantees can make separable problems algorithmically inseparable.

### 3.1 LP-relaxed inference

Consider the simple Markov random field pictured in Figure 1, a triangle in which each node has as its set of allowed labels a different pair of the three possible labels A, B, and C. Let the node potentials $\psi_i(y_i)$ be fixed to 1 so that labeling preferences derive only from edge potentials. For positive

constants $\lambda_{ij}$, define edge potentials $\psi_{ij}(y_i, y_j) = \exp(\lambda_{ij})$ whenever $y_i = y_j$ and $\psi_{ij}(y_i, y_j) = 1$ otherwise. Then the joint probability of a configuration $\mathbf{y} = (y_1, y_2, y_3)$ is given by

$$P(\mathbf{y}) \propto \prod_{ij:y_i=y_j} \exp(\lambda_{ij}) = \exp\left(\sum_{i,j} \mathbb{I}(y_i = y_j)\lambda_{ij}\right) \tag{4}$$

and the MAP labeling is $\arg\max_{\mathbf{y}} \left[\sum_{i,j} \mathbb{I}(y_i = y_j)\lambda_{ij}\right]$.

Note that this example is associative; that is, neighboring nodes are encouraged to take identical labels ($\lambda_{ij} > 0$). We can therefore perform approximate inference using a linear programming (LP) relaxation and get a multiplicative approximation guarantee [3]. We begin by writing an integer program for computing the MAP labeling; below, $\mu_i(y_i)$ indicates node $i$ taking label $y_i$ (which ranges over the two allowed labels for node $i$) and $\mu_{ij}(y_i, y_j)$ indicates nodes $i$ and $j$ taking labels $y_i$ and $y_j$, respectively.

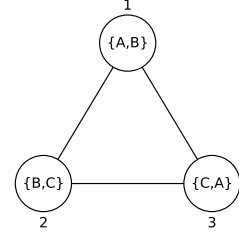

$$\max_{\boldsymbol{\mu}} \quad \lambda_{12}\mu_{12}(B, B) + \lambda_{23}\mu_{23}(C, C) + \lambda_{31}\mu_{31}(A, A)$$

$$\text{s.t.} \quad \sum_{y_i} \mu_i(y_i) \leq 1 \quad \forall i$$

$$\mu_{ij}(y_i, y_j) \leq \mu_i(y_i) \quad \forall ij, y_i, y_j$$

$$\boldsymbol{\mu} \in \{0, 1\}^{\dim(\boldsymbol{\mu})}$$

Figure 1: A simple MRF. Each node is annotated with its allowed labels.

Integer programming is NP-hard, so we use an LP-relaxation by replacing the integrality constraint with $\boldsymbol{\mu} \geq \mathbf{0}$. Letting $i^*j^* = \arg\max_{ij} \lambda_{ij}$, it is easy to see that the correct MAP configuration assigns matching labels to nodes $i^*$ and $j^*$ and an arbitrary label to the third. The score for this configuration is $\lambda_{i^*j^*}$. However, the LP-relaxation may generate fractional solutions. In particular, whenever $(\lambda_{12} + \lambda_{23} + \lambda_{31})/2 > \lambda_{i^*j^*}$ the configuration that assigns to every node both of its allowed labels in equal proportion—$\boldsymbol{\mu} = 1/2$—is optimal.

The fractional labeling $\boldsymbol{\mu} = 1/2$ is the most uninformative possible; it suggests that all labelings are equally valid. Even so, $(\lambda_{12} + \lambda_{23} + \lambda_{31})/2 \leq 3\lambda_{i^*j^*}/2$ by the definition of $i^*j^*$, so LP-relaxed inference for this MRF has a relatively good approximation ratio of $3/2$.

## 3.2  Learning with LP-relaxed inference

Suppose now that we wish to learn to predict labelings $\mathbf{y}$ from instances of the MRF in Figure 1 with positive features given by $\mathbf{x} = (x_{12}, x_{23}, x_{31})$. We will parameterize the model using a positive weight vector $\mathbf{w} = (w_{12}, w_{23}, w_{31})$, letting $\lambda_{ij} = w_{ij}x_{ij}$.

Suppose the data distribution gives equal probability to inputs $\mathbf{x} = (4, 3, 3)$, $(3, 4, 3)$, and $(3, 3, 4)$, and that the loss function is defined as follows. Given $\mathbf{x}$, let $i^*j^* = \arg\max_{ij} x_{ij}$. Then assigning matching labels to nodes $i^*$ and $j^*$ and an arbitrary label to the third node yields a 0-loss configuration. All other configurations have positive loss. It is clear, first of all, that this problem is separable; if $\mathbf{w} = (1, 1, 1)$, $\lambda_{ij} = x_{ij}$ and the solution to the integer program above coincides with the labeling rule. Furthermore, there is margin: any weight vector in a neighborhood of $(1, 1, 1)$ assigns the highest probability to the correct labeling.

Using LP-relaxed inference, however, the problem is impossible to learn. In order to correctly label the instance $\mathbf{x} = (4, 3, 3)$ we must have, at a minimum, $\lambda_{12} > \lambda_{23}, \lambda_{31}$ (equivalently $4w_{12} > 3w_{23}, 3w_{31}$) since the 0-loss labeling must have higher objective score than any other labeling. Reasoning similarly for the remaining instances, any separating weight vector must satisfy $4w_{ij} > 3w_{kl}$ for each pair of edges $(ij, kl)$. Without loss of generality, assume an instance to be labeled has feature vector $\mathbf{x} = (4, 3, 3)$. Then,

$$\begin{aligned}
\frac{1}{2}(\lambda_{12} + \lambda_{23} + \lambda_{31}) &= \frac{1}{2}(4w_{12} + 3w_{23} + 3w_{31}) \\
&> \frac{1}{2}(4w_{12} + 3\frac{3}{4}w_{12} + 3\frac{3}{4}w_{12}) \\
&> 4w_{12} \\
&= \lambda_{12}.
\end{aligned}$$

As a result, LP-relaxed inference predicts $\boldsymbol{\mu} = \mathbf{1/2}$. The data cannot be correctly labeled using an LP-relaxation with *any* choice of weight vector, and the example is therefore algorithmically inseparable.

## 4 Insufficiency of algorithmic separability

We cannot expect to learn without algorithmic separability; no amount of training can hope to be successful when there simply do not exist acceptable model parameters. Nevertheless, we could draw upon the usual techniques for dealing with (geometric) inseparability in this case.

Approximate inference introduces another complication, however. Learning techniques exploit assumptions about the underlying model to search parameter space; the perceptron, for example, assumes that increasing weights for features present in correct labelings but not incorrect labelings will lead to better predictions. While this is formally true with respect to an underlying linear model, inexact inference methods can disturb and even invert such assumptions.

### 4.1 Loopy inference

Loopy belief propagation (LBP) is a common approximate inference procedure in which max-product message passing, known to be exact for trees, is applied to arbitrary, cyclic graphical models [5]. While LBP is, of course, inexact, its behavior can be even more problematic for learning. Because LBP does not respond to model parameters in the usual way, its predictions can lead a learner away from appropriate parameters even for algorithmically separable problems.

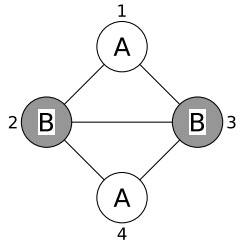

Figure 2: An MRF on which LBP is inexact.

Consider the simple MRF shown in Figure 2 and discussed previously in [6]. All nodes are binary and take labels from the set $\{-1, 1\}$. Suppose that node potentials are assigned by type, where each node is of type $A$ or $B$ as indicated and $\alpha$ and $\beta$ are real-valued parameters:

$$\psi_A(-1) = 1 \qquad\qquad \psi_A(1) = e^\alpha$$
$$\psi_B(-1) = 1 \qquad\qquad \psi_B(1) = e^\beta$$

Also let edge potentials $\psi_{ij}(y_i, y_j)$ be equal to the constant $\lambda$ when $y_i = y_j$ and 1 otherwise. Define $\lambda$ to be sufficiently positive that the MAP configuration is either $(-1, -1, -1, -1)$ or $(1, 1, 1, 1)$, abbreviated by $-\mathbf{1}$ and $\mathbf{1}$, respectively. In particular, the solution is $-\mathbf{1}$ when $\alpha + \beta < 0$ and $\mathbf{1}$ otherwise. With slight abuse of notation we can write $\mathbf{y}_{\text{MAP}} = \text{sign}(\alpha + \beta)$.

We now investigate the behavior of LBP on this example. In general, max-product LBP on pairwise MRFs requires iterating the following rule to update messages $m_{ij}(y_j)$ from node $i$ to node $j$, where $y_j$ ranges over the possible labels for node $j$ and $N(i)$ is the neighbor set of node $i$.

$$m_{ij}(y_j) = \max_{y_i} \left[ \psi_{ij}(y_i, y_j)\psi_i(y_i) \prod_{k \in N(i) \backslash \{j\}} m_{ki}(y_i) \right] \tag{5}$$

Since we take $\lambda$ to be suitably positive in our example, we can eliminate the max, letting $y_i = y_j$, and then divide to remove the edge potentials $\psi_{ij}(y_j, y_j) = \lambda$. When messages are initialized uniformly to 1 and passed in parallel, symmetry also implies that messages are completely determined by the the types of the relevant nodes. The updates are then as follows.

$$m_{AB}(-1) = m_{BA}(-1) \qquad\qquad m_{AB}(1) = e^\alpha m_{BA}(1)$$
$$m_{BA}(-1) = m_{AB}(-1)m_{BB}(-1) \qquad\qquad m_{BA}(1) = e^\beta m_{AB}(1)m_{BB}(1)$$
$$m_{BB}(-1) = m_{AB}^2(-1) \qquad\qquad m_{BB}(1) = e^\beta m_{AB}^2(1)$$

Note that messages $m_{ij}(-1)$ remain fixed at 1 after any number of updates. Messages $m_{AB}(1)$, $m_{BA}(1)$, and $m_{BB}(1)$ always take the form $\exp(p\alpha + q\beta)$ for appropriate values of $p$ and $q$, and it is easy to show by iterating the updates that, for all three messages, $p$ and $q$ go to $\infty$ while the ratio $q/p$ converges to $\gamma \approx 1.089339$. The label 1 messages, therefore, approach 0 when $\alpha + \gamma\beta < 0$ and $\infty$ when $\alpha + \gamma\beta > 0$. Note that after message normalization ($m_{ij}(-1) + m_{ij}(1) = 1$ for all $ij$) the algorithm converges in either case.

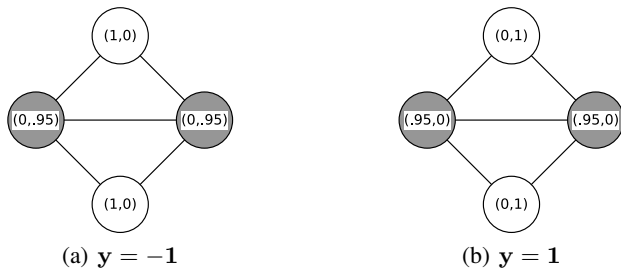

(a) $\mathbf{y} = -\mathbf{1}$                    (b) $\mathbf{y} = \mathbf{1}$

Figure 3: A two-instance training set. Within each instance, nodes of the same shading share a feature vector, as annotated. Below each instance is its correct labeling.

Beliefs are computed from the converged messages as $b_i(y_i) \propto \prod_{j \in N(i)} m_{ji}(y_i)$, so we can express the prediction of LBP as $\mathbf{y}_{\mathrm{LBP}} = \mathrm{sign}(\alpha + \gamma\beta)$. Intuitively, then, LBP gives a slight preference to the $B$-type nodes because of their shared edge. If $\alpha$ and $\beta$ are both positive or both negative, or if $\alpha$ and $\beta$ differ in sign but $|\beta| > |\alpha|$ or $|\alpha| > \gamma|\beta|$, LBP finds the correct MAP solution. However, when the strength of the $A$ nodes only slightly exceeds that of the $B$ nodes ($\gamma|\beta| > |\alpha| > |\beta|$), the preference exerted by LBP is significant enough to flip the labels. For example, if $\alpha = 1$ and $\beta = -0.95$, the true MAP configuration is $\mathbf{1}$ but LBP converges to $-\mathbf{1}$.

## 4.2 Learning with LBP

Suppose now that we wish to use the perceptron algorithm with LBP inference to learn the two-instance data set shown in Figure 3. For each instance the unshaded nodes are annotated with a feature vector $\mathbf{x}_\alpha = (x_{\alpha 1}, x_{\alpha 2})$ and the shaded nodes are annotated with a feature vector $\mathbf{x}_\beta = (x_{\beta 1}, x_{\beta 2})$. We wish to learn weights $\mathbf{w} = (w_1, w_2)$, modeling node potentials as before with $\alpha = \mathbf{w} \cdot \mathbf{x}_\alpha$ and $\beta = \mathbf{w} \cdot \mathbf{x}_\beta$. Assume that edge potentials remain fixed using a suitably positive $\lambda$.

By the previous analysis, the data are algorithmically separated by $\mathbf{w}^* = (1, -1)$. On instance (a), $\alpha = 1$, $\beta = -0.95$, and LBP correctly predicts $-\mathbf{1}$. Instance (b) is symmetric. Note that although the predicted configurations are not the true MAP labelings, they correctly match the training labels. The weight vector $(1, -1)$ is therefore an ideal choice in the context of learning. The problem is also separated in the usual sense by the weight vector $(-1, 1)$.

Since we can think of the MAP decision problem as computing $\mathrm{sign}(\alpha + \beta) = \mathrm{sign}\,(\mathbf{w} \cdot (\mathbf{x}_\alpha + \mathbf{x}_\beta))$, we can apply the perceptron algorithm with update $\mathbf{w} \leftarrow \mathbf{w} - \hat{y}(\mathbf{x}_\alpha + \mathbf{x}_\beta)$, where $\hat{y}$ is the sign of the proposed labeling. The standard perceptron mistake bound guarantees that separable problems require only a finite number of iterations with exact inference to find a separating weight vector. Here, however, LBP causes the perceptron to diverge even though the problem is not only separable but also algorithmically separable.

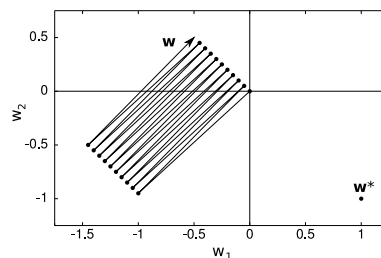

Figure 4: Perceptron learning path.

Figure 4 shows the path of the weight vector as it progresses from the origin over the first 20 iterations of the algorithm. During each pass through the data the weight vector is updated twice: once after mislabeling instance (a) ($\mathbf{w} \leftarrow \mathbf{w} - (1, 0.95)$), and again after mislabeling instance (b) ($\mathbf{w} \leftarrow \mathbf{w} + (0.95, 1)$). The net effect is $\mathbf{w} \leftarrow \mathbf{w} + (-0.05, 0.05)$. The weight vector continually moves in the opposite direction of $\mathbf{w}^* = (1, -1)$, and learning diverges.

## 4.3 Discussion

To understand why perceptron learning fails with LBP, it is instructive to visualize the feasible regions of weight space. Exact inference correctly labels instance (a) whenever $w_1 + 0.95w_2 < 0$, and, similarly, instance (b) requires a weight vector with $0.95w_1 + w2 > 0$. Weights that satisfy both constraints are feasible, as depicted in Figure 5(a). For LBP, the preference given to nodes 2 and 3 is effectively a scaling of $\mathbf{x}_\beta$ by $\gamma \approx 1.089339$, so a feasible weight vector must satisfy

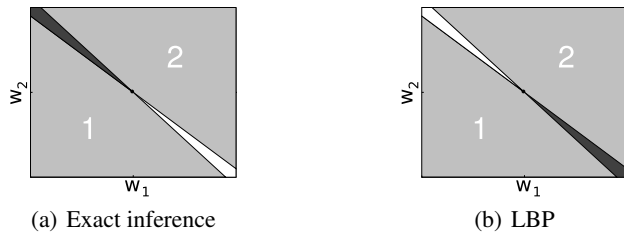

(a) Exact inference          (b) LBP

Figure 5: The feasible regions of weight space for exact inference and LBP. Each numbered gray halfspace indicates the region in which the corresponding instance is correctly labeled; their intersection is the feasible region, colored black.

$w_1 + 0.95\gamma w_2 < 0$ and $0.95\gamma w_1 + w_2 > 0$. Since $0.95\gamma > 1$, these constraints define a completely different feasible region of weight space, shown in Figure 5(b). It is clear from the figures why perceptron does not succeed; it assumes that pushing weights into the feasible region of Figure 5(a) will produce correct labelings, while under LBP the exact opposite is required.

Algorithmic separability, then, is necessary for learning but may not be sufficient. This does not imply that no algorithm can learn using LBP; a grid search on weight space, for example, will be slow but successful. Instead, care must be taken to ensure that learning and inference are appropriately matched. In particular, it is generally invalid to assume that an arbitrary choice of approximate inference will lead to useful results when the learning method expects exact feedback.

## 5 Learning bounds for approximate inference

In contrast to the failure of LBP in Section 4, appropriate pairs of inference and learning algorithms do exist. We give two bounds using LP-relaxed inference for MRFs with log-linear potentials. First, under the assumption of algorithmic separability, we show that the structured perceptron of Collins [9] makes only a finite number of mistakes. Second, we show using the PAC-Bayesian framework [11] that choosing model parameters to minimize a margin-based empirical risk function (assuming "soft" algorithmic separability) gives rise to a bound on the true risk. In both cases, the proofs are directly adapted from known results using the following characterization of LP-relaxation.

**Claim 1.** *Let $\mathbf{z} = (z_1, \ldots, z_k)$ be the vector of 0/1 optimization variables for an integer program $P$. Let $\mathcal{Z} \subseteq \{0,1\}^{\dim(\mathbf{z})}$ be the feasible set of $P$. Then replacing integrality constraints in $P$ with box constraints $0 \leq z_i \leq 1$ yields an LP with a feasible polytope having vertices $\mathcal{Z}' \supseteq \mathcal{Z}$.*

*Proof.* Each $\mathbf{z} \in \mathcal{Z}$ is integral and thus a vertex of the polytope defined by box constraints alone. The remaining constraints appear in $P$ and by definition do not exclude any element of $\mathcal{Z}$. The addition of constraints cannot eliminate a vertex without rendering it infeasible. Thus, $\mathcal{Z} \subseteq \mathcal{Z}'$. □

We can encode the MAP inference problem for MRFs as an integer program over indicators $\mathbf{z}$ with objective $\mathbf{w} \cdot \Phi(\mathbf{x}, \mathbf{z})$ for some $\Phi$ linear in $\mathbf{z}$ (see, for example, [6]). By Claim 1 and the fact that an optimal vertex always exists, LP-relaxed inference given an input $\mathbf{x}$ computes

$$\mathrm{LP}_{\mathbf{w}}(\mathbf{x}) = \arg\max_{\mathbf{z} \in \mathcal{Z}'(\mathbf{x})} \mathbf{w} \cdot \Phi(\mathbf{x}, \mathbf{z}) . \tag{6}$$

We can think of this as exact inference over an expanded set of labelings $\mathcal{Z}'(\mathbf{x})$, some of which may not be valid (i.e., $\mathbf{z} \in \mathcal{Z}'(\mathbf{x})$ may be fractional). To simplify notation, we will assume that labelings $\mathbf{y}$ are always translated into corresponding indicator values $\mathbf{z}$.

### 5.1 Perceptron

**Theorem 1** (adapted from Theorem 1 in [9]). *Given a sequence of input/labeling pairs $\{(\mathbf{x}^i, \mathbf{z}^i)\}$, suppose that there exists a weight vector $\mathbf{w}^*$ with unit norm and $\gamma > 0$ such that, for all $i$, $\mathbf{w}^* \cdot (\Phi(\mathbf{x}^i, \mathbf{z}^i) - \Phi(\mathbf{x}^i, \mathbf{z})) \geq \gamma$ for all $\mathbf{z} \in \mathcal{Z}'(\mathbf{x}^i) \setminus \{\mathbf{z}^i\}$. (The instances are algorithmically separable with margin $\gamma$.) Suppose that there also exists $R$ such that $\|\Phi(\mathbf{x}^i, \mathbf{z}^i) - \Phi(\mathbf{x}^i, \mathbf{z})\| \leq R$ for all $\mathbf{z} \in \mathcal{Z}'(\mathbf{x}^i)$. Then the structured perceptron makes at most $R^2/\gamma^2$ mistakes.*

*Proof sketch.* Let $\mathbf{w}^k$ be the weight vector before the $k$th mistake; $\mathbf{w}^1 = \mathbf{0}$. Following the proof of Collins without modification, we can show that $\|\mathbf{w}^{k+1}\| \geq k\gamma$. We now bound $\|\mathbf{w}^{k+1}\|$ in the other direction. If $(\mathbf{x}_k, \mathbf{z}_k)$ is the instance on which the $k$th update occurs and $\mathbf{z}_{\mathrm{LP}(k)} = \mathrm{LP}_{\mathbf{w}^k}(\mathbf{x}_k)$, then by the update rule,

$$
\begin{aligned}
\|\mathbf{w}^{k+1}\|^2 &= \|\mathbf{w}^k\|^2 + 2\mathbf{w}^k \cdot (\Phi(\mathbf{x}_k, \mathbf{z}_k) - \Phi(\mathbf{x}_k, \mathbf{z}_{\mathrm{LP}(k)})) + \|\Phi(\mathbf{x}_k, \mathbf{z}_k) - \Phi(\mathbf{x}_k, \mathbf{z}_{\mathrm{LP}(k)})\|^2 \\
&\leq \|\mathbf{w}^k\|^2 + R^2 .
\end{aligned}
\tag{7}
$$

The inequality follows from the fact that LP-relaxed inference maximizes $\mathbf{w} \cdot \Phi(\mathbf{x}_k, \mathbf{z})$ over all $\mathbf{z} \in \mathcal{Z}'(\mathbf{x}_k)$, so the middle term is nonpositive. Hence, by induction, $\|\mathbf{w}^{k+1}\|^2 \leq kR^2$. Combining the two bounds, $k^2\gamma^2 \leq \|\mathbf{w}^{k+1}\|^2 \leq kR^2$, hence $k \leq R^2/\gamma^2$. $\qquad\square$

## 5.2 PAC-Bayes

The perceptron bound applies when data are perfectly algorithmically separable, but we might also hope to use LP-relaxed inference in the presence of noisy or otherwise almost-separable data. The following theorem adapts an empirical risk minimization bound using the PAC-Bayes framework to show that LP-relaxed inference can also be used to learn successfully in these cases. The measure of empirical risk for a weight vector $\mathbf{w}$ over a sample $S = (\mathbf{x}^1, \dots, \mathbf{x}^m)$ is defined as follows.

$$
\hat{\mathcal{R}}(\mathbf{w}, S) = \frac{1}{m} \sum_{i=1}^m \max_{\mathbf{z} \in H_{\mathbf{w}}(\mathbf{x}^i)} \mathcal{L}(\mathbf{z}|\mathbf{x}^i)
\tag{8}
$$
$$
H_{\mathbf{w}}(\mathbf{x}) = \{\mathbf{z}' \in \mathcal{Z}'(\mathbf{x}) \mid \mathbf{w} \cdot (\Phi(\mathbf{x}, \mathrm{LP}_{\mathbf{w}}(\mathbf{x})) - \Phi(\mathbf{x}, \mathbf{z}')) \leq |\mathrm{LP}_{\mathbf{w}}(\mathbf{x}) - \mathbf{z}'|\}
$$

Intuitively, $\hat{\mathcal{R}}$ accounts for the maximum loss of any $\mathbf{z}$ that is closer in score than in 1-norm to the LP prediction. Such $\mathbf{z}$ are considered "confusable" at test time. The PAC-Bayesian setting requires that, after training, weight vectors are drawn from some distribution $Q(\mathbf{w})$; however, a deterministic version of the bound can also be proved.

**Theorem 2** (adapted from Theorem 3 in [11]). *Suppose that loss function $\mathcal{L}(\mathbf{y}|\mathbf{x})$ is bounded between 0 and 1 and can be expanded to $\mathcal{L}(\mathbf{z}|\mathbf{x})$ for all $\mathbf{z} \in \mathcal{Z}'(\mathbf{x})$; that is, loss can be defined for every potential value of $\mathrm{LP}(\mathbf{x})$. Let $\ell = \dim(\mathbf{z})$ be the number of indicator variables in the LP, and let $R$ bound the 2-norm of a feature vector for a single clique. Let $Q(\mathbf{w})$ be a symmetric Gaussian centered at $\mathbf{w}$ as defined in [11]. Then with probability at least $1 - \delta$ over the choice of a sample $S$ of size $m$ from distribution $D$ over inputs $\mathbf{x}$, the following holds for all $\mathbf{w}$.*

$$
E_{\mathbf{x} \sim D, \mathbf{w}' \sim Q(\mathbf{w})} \left[\mathcal{L}(\mathrm{LP}_{\mathbf{w}'}(\mathbf{x})|\mathbf{x})\right] \leq \hat{\mathcal{R}}(\mathbf{w}, S) + \sqrt{\frac{R^2\|\mathbf{w}\|^2 \ln(\frac{2\ell m}{R^2\|w\|^2}) + \ln(\frac{m}{\delta})}{2(m-1)}} + \frac{R^2\|\mathbf{w}\|^2}{m}
\tag{9}
$$

The proof in [11] can be directly adapted; the only significant changes are the use of $\mathcal{Z}'$ in place of the set $\mathcal{Y}$ of possible labelings and reasoning as above using the definition of LP-relaxed inference.

## 6  Related work

A number of authors have applied inference approximations to a wide range of learning problems, sometimes with theoretical analysis of approximation quality and often with good empirical results [8, 12, 3]. However, none to our knowledge has investigated the theoretical relationship between approximation and learning performance. Daume et al. [13] developed a method for using a linear model to make decisions during a search-based approximate inference process. They showed that perceptron updates give rise to a mistake bound under the assumption that parameters leading to correct decisions exist. Such results are analogous to those presented in Section 5 in that performance bounds follow from an (implicit) assumption of algorithmic separability.

Wainright [14] proved that when approximate inference is required at test time due to computational constraints, using an inconsistent (approximate) estimator for learning can be beneficial. His result suggests that optimal performance is obtained when the methods used for training and testing are appropriately aligned, even if those methods are not independently optimal. In contrast, we consider learning algorithms that use identical inference for both training and testing, minimizing a general measure of empirical risk rather than maximizing data likelihood, and argue for compatibility between the learning method and inference process.

Roth et al. [15] consider learning independent classifiers for single labels, essentially using a trivial form of approximate inference. They show that this method can outperform exact inference learning when algorithmic separability holds precisely *because* approximation reduces expressivity; i.e., less complex models require fewer samples to train accurately. When the data are not algorithmically separable, exact inference provides better performance if a large enough sample is available. It is interesting to note that both of our counterexamples involve strong edge potentials. These are precisely the kinds of examples that are difficult to learn using independent classifiers.

## 7 Conclusion

Effective use of approximate inference for learning depends on two considerations that are irrelevant for prediction. First, the expressivity of approximate inference, and consequently the bias for learning, can vary significantly from that of exact inference. Second, learning algorithms can misinterpret feedback received from approximate inference methods, leading to poor results or even divergence. However, when algorithmic separability holds, the use of LP-relaxed inference with standard learning frameworks yields provably good results.

Future work includes the investigation of alternate inference methods that, while potentially less suitable for prediction alone, give better feedback for learning. Conversely, learning methods that are tailored specifically to particular inference algorithms might show improved performance over those that assume exact inference. Finally, the notion of algorithmic separability and the ways in which it might relate (through approximation) to traditional separability deserve further study.

## Footnotes

[1] Separability can be weakened to allow nonzero risk, but for simplicity we focus on the strict case.

[2] Note further that algorithmic separability supports inference algorithms that are not based on any abstract model at all; such algorithms can describe arbitrary "black box" functions from parameters to predictions. It seems unlikely, however, that such algorithms are of much use since their parameters cannot be easily learned.

## References

[1] Gregory F. Cooper. The computational complexity of probabilistic inference using Bayesian belief networks (research note). *Artif. Intell.*, 42(2-3):393–405, 1990.

[2] John D. Lafferty, Andrew McCallum, and Fernando C. N. Pereira. Conditional random fields: Probabilistic models for segmenting and labeling sequence data. In *ICML '01: Proceedings of the Eighteenth International Conference on Machine Learning*, pages 282–289, 2001.

[3] Ben Taskar, Vassil Chatalbashev, and Daphne Koller. Learning associative Markov networks. In *ICML '04: Proceedings of the twenty-first international conference on Machine learning*, page 102, 2004.

[4] Judea Pearl. *Probabilistic reasoning in intelligent systems: networks of plausible inference*. Morgan Kaufmann Publishers Inc., San Francisco, CA, USA, 1988.

[5] Kevin Murphy, Yair Weiss, and Michael Jordan. Loopy belief propagation for approximate inference: An empirical study. In *Proceedings of the 15th Annual Conference on Uncertainty in Artificial Intelligence (UAI-99)*, pages 467–47, 1999.

[6] M.J. Wainwright, T.S. Jaakkola, and A.S. Willsky. MAP estimation via agreement on trees: message-passing and linear programming. *IEEE Transactions on Information Theory*, 51(11):3697–3717, 2005.

[7] D. Roth and W. Yih. A linear programming formulation for global inference in natural language tasks. In *Proc. of the Conference on Computational Natural Language Learning (CoNLL)*, pages 1–8, 2004.

[8] Charles Sutton and Andrew McCallum. Collective segmentation and labeling of distant entities in information extraction. Technical Report TR # 04-49, University of Massachusetts, 2004.

[9] Michael Collins. Discriminative training methods for hidden Markov models: theory and experiments with perceptron algorithms. In *EMNLP '02: Proceedings of the ACL-02 conference on Empirical methods in natural language processing*, pages 1–8, 2002.

[10] David A. McAllester. PAC-bayesian stochastic model selection. *Machine Learning*, 51(1):5–21, 2003.

[11] David McAllester. Generalization bounds and consistency for structured labeling. In *Predicting Structured Data*. MIT Press, To Appear.

[12] Charles Sutton and Andrew McCallum. Piecewise training of undirected models. In *21st Conference on Uncertainty in Artificial Intelligence*, 2005.

[13] Hal Daumé III and Daniel Marcu. Learning as search optimization: Approximate large margin methods for structured prediction. In *International Conference on Machine Learning (ICML)*, 2005.

[14] Martin J. Wainwright. Estimating the "wrong" graphical model: Benefits in the computation-limited setting. *Journal of Machine Learning Research*, 7:1829–1859, 2006.

[15] V. Punyakanok, D. Roth, W. Yih, and D. Zimak. Learning and inference over constrained output. In *Proc. of the International Joint Conference on Artificial Intelligence (IJCAI)*, pages 1124–1129, 2005.

